# Sparse and Locally Constant Gaussian Graphical Models

**Jean Honorio,    Luis Ortiz,    Dimitris Samaras**
Department of Computer Science
Stony Brook University
Stony Brook, NY 11794
{jhonorio,leortiz,samaras}@cs.sunysb.edu

**Nikos Paragios**
Laboratoire MAS
Ecole Centrale Paris
Chatenay-Malabry, France
nikos.paragios@ecp.fr

**Rita Goldstein**
Medical Department
Brookhaven National Laboratory
Upton, NY 11973
rgoldstein@bnl.gov

## Abstract

Locality information is crucial in datasets where each variable corresponds to a measurement in a manifold (silhouettes, motion trajectories, 2D and 3D images). Although these datasets are typically under-sampled and high-dimensional, they often need to be represented with low-complexity statistical models, which are comprised of only the important probabilistic dependencies in the datasets. Most methods attempt to reduce model complexity by enforcing structure sparseness. However, sparseness cannot describe inherent regularities in the structure. Hence, in this paper we first propose a new class of Gaussian graphical models which, together with sparseness, imposes local constancy through $\ell_1$-norm penalization. Second, we propose an efficient algorithm which decomposes the strictly convex maximum likelihood estimation into a sequence of problems with closed form solutions. Through synthetic experiments, we evaluate the closeness of the recovered models to the ground truth. We also test the generalization performance of our method in a wide range of complex real-world datasets and demonstrate that it captures useful structures such as the rotation and shrinking of a beating heart, motion correlations between body parts during walking and functional interactions of brain regions. Our method outperforms the state-of-the-art structure learning techniques for Gaussian graphical models both for small and large datasets.

## 1  Introduction

Structure learning aims to discover the topology of a probabilistic network of variables such that this network represents accurately a given dataset while maintaining low complexity. Accuracy of representation is measured by the likelihood that the model explains the observed data, while complexity of a graphical model is measured by its number of parameters. Structure learning faces several challenges: the number of possible structures is super-exponential in the number of variables while the required sample size might be even exponential. Therefore, finding good regularization techniques is very important in order to avoid over-fitting and to achieve a better generalization performance. In this paper, we propose *local constancy* as a prior for learning Gaussian graphical models, which is natural for spatial datasets such as those encountered in computer vision [1, 2, 3].

For Gaussian graphical models, the number of parameters, the number of edges in the structure and the number of non-zero elements in the inverse covariance or precision matrix are equivalent

measures of complexity. Therefore, several techniques focus on enforcing sparsity of the precision matrix. An approximation method proposed in [4] relied on a sequence of sparse regressions. Maximum likelihood estimation with an $\ell_1$-norm penalty for encouraging sparseness is proposed in [5, 6, 7]. The difference among those methods is the optimization technique: a sequence of box-constrained quadratic programs in [5], solution of the dual problem by sparse regression in [6] or an approximation via standard determinant maximization with linear inequality constraints in [7]. It has been shown theoretically and experimentally, that only the *covariance selection* [5] as well as *graphical lasso* [6] converge to the maximum likelihood estimator.

In datasets which are a collection of measurements for variables with some spatial arrangement, one can define a local neighborhood for each variable or manifold. Such variables correspond to points in silhouettes, pixels in 2D images or voxels in 3D images. Silhouettes define a natural one-dimensional neighborhood in which each point has two neighbors on each side of the closed contour. Similarly, one can define a four-pixel neighborhood for 2D images as well as six-pixel neighborhood for 3D images. However, there is little research on spatial regularization for structure learning. Some methods assume a one-dimensional spatial neighborhood (e.g. silhouettes) and that variables far apart are only weakly correlated [8], interaction between a priori known groups of variables as in [9], or block structures as in [10] in the context of Bayesian networks.

Our contribution in this paper is two-fold. First, we propose *local constancy*, which encourages finding connectivities between two close or distant clusters of variables, instead of between isolated variables. It does not heavily constrain the set of possible structures, since it only imposes restrictions of spatial closeness for each cluster independently, but not between clusters. We impose an $\ell_1$-norm penalty for differences of spatially neighboring variables, which allows obtaining locally constant models that preserve sparseness, unlike $\ell_2$-norm penalties. Our model is strictly convex and therefore has a global minimum. Positive definiteness of the estimated precision matrix is also guaranteed, since this is a necessary condition for the definition of a multivariate normal distribution.

Second, since optimization methods for structure learning on Gaussian graphical models [5, 6, 4, 7] are unable to handle local constancy constraints, we propose an efficient algorithm by maximizing with respect to one row and column of the precision matrix at a time. By taking directions involving either one variable or two spatially neighboring variables, the problem reduces to minimization of a piecewise quadratic function, which can be performed in closed form.

We initially test the ability of our method to recover the ground truth structure from data, of a complex synthetic model which includes locally and not locally constant interactions as well as independent variables. Our method outperforms the state-of-the-art structure learning techniques [5, 6, 4] for datasets with both small and large number of samples. We further show that our method has better generalization performance on real-world datasets. We demonstrate the ability of our method to discover useful structures from datasets with a diverse nature of probabilistic relationships and spatial neighborhoods: manually labeled silhouettes in a walking sequence, cardiac magnetic resonance images (MRI) and functional brain MRI.

Section 2 introduces Gaussian graphical models as well as techniques for learning such structures from data. Section 3 presents our sparse and locally constant Gaussian graphical models. Section 4 describes our structure learning algorithm. Experimental results on synthetic and real-world datasets are shown and explained in Section 5. Main contributions and results are summarized in Section 6.

## 2 Background

In this paper, we use the notation in Table 1. For convenience, we define two new operators: the zero structure operator and the diagonal excluded product.

A *Gaussian graphical model* [11] is a graph in which all random variables are continuous and jointly Gaussian. This model corresponds to the multivariate normal distribution for $N$ variables $\mathbf{x} \in \mathbb{R}^N$ with mean vector $\boldsymbol{\mu} \in \mathbb{R}^N$ and a covariance matrix $\boldsymbol{\Sigma} \in \mathbb{R}^{N \times N}$, or equivalently $\mathbf{x} \sim \mathcal{N}(\boldsymbol{\mu}, \boldsymbol{\Sigma})$ where $\boldsymbol{\Sigma} \succ \mathbf{0}$. Conditional independence in a Gaussian graphical model is simply reflected in the zero entries of the precision matrix $\boldsymbol{\Omega} = \boldsymbol{\Sigma}^{-1}$ [11]. Let $\boldsymbol{\Omega} = \{\omega_{n_1 n_2}\}$, two variables $x_{n_1}$ and $x_{n_2}$ are conditionally independent if and only if $\omega_{n_1 n_2} = 0$. The precision matrix representation is preferred because it allows detecting cases in which two seemingly correlated variables, actually depend on a third confounding variable.

| Notation | Description |
|---|---|
| $\|\mathbf{c}\|_1$ | $\ell_1$-norm of $\mathbf{c} \in \mathbb{R}^N$, i.e. $\sum_n |c_n|$ |
| $\|\mathbf{c}\|_\infty$ | $\ell_\infty$-norm of $\mathbf{c} \in \mathbb{R}^N$, i.e. $\max_n |c_n|$ |
| $|\mathbf{c}|$ | entrywise absolute value of $\mathbf{c} \in \mathbb{R}^N$, i.e. $(|c_1|, |c_2|, \ldots, |c_N|)^{\mathrm{T}}$ |
| $\mathbf{diag}(\mathbf{c}) \in \mathbb{R}^{N \times N}$ | matrix with elements of $\mathbf{c} \in \mathbb{R}^N$ on its diagonal |
| $\|\mathbf{A}\|_1$ | $\ell_1$-norm of $\mathbf{A} \in \mathbb{R}^{M \times N}$, i.e. $\sum_{mn} |a_{mn}|$ |
| $\langle \mathbf{A}, \mathbf{B} \rangle$ | scalar product of $\mathbf{A}, \mathbf{B} \in \mathbb{R}^{M \times N}$, i.e. $\sum_{mn} a_{mn} b_{mn}$ |
| $\mathbf{A} \circ \mathbf{B} \in \mathbb{R}^{M \times N}$ | Hadamard or entrywise product of $\mathbf{A}, \mathbf{B} \in \mathbb{R}^{M \times N}$, i.e. $(\mathbf{A} \circ \mathbf{B})_{mn} = a_{mn} b_{mn}$ |
| $\mathbf{J}(\mathbf{A}) \in \mathbb{R}^{M \times N}$ | *zero structure operator* of $\mathbf{A} \in \mathbb{R}^{M \times N}$, by using the Iverson bracket $j_{mn}(\mathbf{A}) = [a_{mn} = 0]$ |
| $\mathbf{A} \oslash \mathbf{B} \in \mathbb{R}^{M \times N}$ | *diagonal excluded product* of $\mathbf{A} \in \mathbb{R}^{M \times N}$ and $\mathbf{B} \in \mathbb{R}^{N \times N}$, i.e. $\mathbf{A} \oslash \mathbf{B} = \mathbf{J}(\mathbf{A}) \circ (\mathbf{AB})$. It has the property that no diagonal entry of $\mathbf{B}$ is used in $\mathbf{A} \oslash \mathbf{B}$ |
| $\mathbf{A} \succ \mathbf{0}$ | $\mathbf{A} \in \mathbb{R}^{N \times N}$ is symmetric and positive definite |
| $\mathbf{diag}(\mathbf{A}) \in \mathbb{R}^{N \times N}$ | matrix with diagonal elements of $\mathbf{A} \in \mathbb{R}^{N \times N}$ only |
| $\mathbf{vec}(\mathbf{A}) \in \mathbb{R}^{MN}$ | vector containing all elements of $\mathbf{A} \in \mathbb{R}^{M \times N}$ |

Table 1: Notation used in this paper.

The concept of robust estimation by performing covariance selection was first introduced in [12] where the number of parameters to be estimated is reduced by setting some elements of the precision matrix $\mathbf{\Omega}$ to zero. Since finding the most sparse precision matrix which fits a dataset is a NP-hard problem [5], in order to overcome it, several $\ell_1$-regularization methods have been proposed for learning Gaussian graphical models from data.

*Covariance selection* [5] starts with a dense sample covariance matrix $\widehat{\mathbf{\Sigma}}$ and fits a sparse precision matrix $\mathbf{\Omega}$ by solving a maximum likelihood estimation problem with a $\ell_1$-norm penalty which encourages sparseness of the precision matrix or conditional independence among variables:

$$\max_{\mathbf{\Omega} \succ \mathbf{0}} \left( \log \det \mathbf{\Omega} - \langle \widehat{\mathbf{\Sigma}}, \mathbf{\Omega} \rangle - \rho \|\mathbf{\Omega}\|_1 \right) \tag{1}$$

for some $\rho > 0$. Covariance selection computes small perturbations on the sample covariance matrix such that it generates a sparse precision matrix, which results in a box-constrained quadratic programming. This method has moderate run time.

The *Meinshausen-Bühlmann approximation* [4] obtains the conditional dependencies by performing a sparse linear regression for each variable, by using *lasso* regression [13]. This method is very fast but does not yield good estimates for lightly regularized models, as noted in [6]. The constrained optimization version of eq.(1) is solved in [7] by applying a standard determinant maximization with linear inequality constraints, which requires iterative linearization of $\|\mathbf{\Omega}\|_1$. This technique in general does not yield the maximum likelihood estimator, as noted in [14]. The *graphical lasso* technique [6] solves the dual form of eq.(1), which results in a lasso regression problem. This method has run times comparable to [4] without sacrificing accuracy in the maximum likelihood estimator.

Structure learning through $\ell_1$-regularization has been also proposed for different types of graphical models: Markov random fields (MRFs) by a clique selection heuristic and approximate inference [15]; Bayesian networks on binary variables by logistic regression [16]; Conditional random fields by pseudo-likelihood and block regularization in order to penalize all parameters of an edge simultaneously [17]; and Ising models, i.e. MRFs on binary variables with pairwise interactions, by logistic regression [18] which is similar in spirit to [4].

There is little work on spatial regularization for structure learning. Adaptive banding on the Cholesky factors of the precision matrix has been proposed in [8]. Instead of using the traditional lasso penalty, a nested lasso penalty is enforced. Entries at the right end of each row are promoted to zero faster than entries close to the diagonal. The main drawback of this technique is the assumption that the more far apart two variables are the more likely they are to be independent. Grouping of entries in the precision matrix into disjoint subsets has been proposed in [9]. Such subsets can model for instance dependencies between different groups of variables in the case of block structures. Although such a formulation allows for more general settings, its main disadvantage is the need for an a priori segmentation of the entries in the precision matrix.

Related approaches have been proposed for Bayesian networks. In [10] it is assumed that variables belong to unknown classes and probabilities of having edges among different classes were enforced to account for structure regularity, thus producing block structures only.

## 3 Sparse and Locally Constant Gaussian Graphical Models

First, we describe our local constancy assumption and its use to model the spatial coherence of dependence/independence relationships. *Local constancy* is defined as follows: if variable $x_{n_1}$ is dependent (or independent) of variable $x_{n_2}$, then a spatial neighbor $x_{n_1'}$ of $x_{n_1}$ is more likely to be dependent (or independent) of $x_{n_2}$. This encourages finding connectivities between two close or distant clusters of variables, instead of between isolated variables. Note that local constancy imposes restrictions of spatial closeness for each cluster independently, but not between clusters.

In this paper, we impose constraints on the difference of entries in the precision matrix $\boldsymbol{\Omega} \in \mathbb{R}^{N \times N}$ for $N$ variables, which correspond to spatially neighboring variables. Let $\widehat{\boldsymbol{\Sigma}} \in \mathbb{R}^{N \times N}$ be the dense sample covariance matrix and $\mathbf{D} \in \mathbb{R}^{M \times N}$ be the discrete derivative operator on the manifold, where $M \in O(N)$ is the number of spatial neighborhood relationships. For instance, in a 2D image, $M$ is the number of pixel pairs that are spatial neighbors on the manifold. More specifically, if pixel $n_1$ and pixel $n_2$ are spatial neighbors, we include a row $m$ in $\mathbf{D}$ such that $d_{mn_1} = 1$, $d_{mn_2} = -1$ and $d_{mn_3} = 0$ for $n_3 \notin \{n_1, n_2\}$. The following penalized maximum likelihood estimation is proposed:

$$\max_{\boldsymbol{\Omega} \succ \mathbf{0}} \left( \log \det \boldsymbol{\Omega} - \langle \widehat{\boldsymbol{\Sigma}}, \boldsymbol{\Omega} \rangle - \rho \|\boldsymbol{\Omega}\|_1 - \tau \|\mathbf{D} \oslash \boldsymbol{\Omega}\|_1 \right) \tag{2}$$

for some $\rho, \tau > 0$. The first two terms model the quality of the fit of the estimated multivariate normal distribution to the dataset. The third term $\rho \|\boldsymbol{\Omega}\|_1$ encourages sparseness while the fourth term $\tau \|\mathbf{D} \oslash \boldsymbol{\Omega}\|_1$ encourages local constancy in the precision matrix by penalizing the differences of spatially neighboring variables.

In conjunction with the $\ell_1$-norm penalty for sparseness, we introduce an $\ell_1$-norm penalty for local constancy. As discussed further in [19], $\ell_1$-norm penalties lead to locally constant models which preserve sparseness, where as $\ell_2$-norm penalties of differences fail to do so.

The use of the diagonal excluded product for penalizing differences instead of the regular product of matrices, is crucial. The regular product of matrices would penalize the difference between the diagonal and off-diagonal entries of the precision matrix, and potentially destroy positive definiteness of the solution for strongly regularized models.

Even though the choice of the linear operator in eq.(2) does not affect the positive definiteness properties of the estimated precision matrix or the optimization algorithm, in the following Section 4, we discuss positive definiteness properties and develop an optimization algorithm for the specific case of the discrete derivative operator $\mathbf{D}$.

## 4 Coordinate-Direction Descent Algorithm

Positive definiteness of the precision matrix is a necessary condition for the definition of a multivariate normal distribution. Furthermore, strict convexity is a very desirable property in optimization, since it ensures the existence of a unique global minimum. Notice that the penalized maximum likelihood estimation problem in eq.(2) is strictly convex due to the convexity properties of $\log \det \boldsymbol{\Omega}$ on the space of symmetric positive definite matrices [20]. Maximization can be performed with respect to one row and column of the precision matrix $\boldsymbol{\Omega}$ at a time. Without loss of generality, we use the last row and column in our derivation, since permutation of rows and columns is always possible. Also, note that rows in $\mathbf{D}$ can be freely permuted without affecting the objective function. Let:

$$\boldsymbol{\Omega} = \begin{bmatrix} \mathbf{W} & \mathbf{y} \\ \mathbf{y}^{\mathrm{T}} & z \end{bmatrix} \quad , \quad \widehat{\boldsymbol{\Sigma}} = \begin{bmatrix} \mathbf{S} & \mathbf{u} \\ \mathbf{u}^{\mathrm{T}} & v \end{bmatrix} \quad , \quad \mathbf{D} = \begin{bmatrix} \mathbf{D}_1 & \mathbf{0}_{M-L} \\ \mathbf{D}_2 & \mathbf{d}_3 \end{bmatrix} \tag{3}$$

where $\mathbf{W}, \mathbf{S} \in \mathbb{R}^{N-1 \times N-1}$, $\mathbf{y}, \mathbf{u} \in \mathbb{R}^{N-1}$, $\mathbf{d}_3 \in \mathbb{R}^L$ is a vector with all entries different than zero, which requires a permutation of rows in $\mathbf{D}$, $\mathbf{D}_1 \in \mathbb{R}^{M-L \times N-1}$ and $\mathbf{D}_2 \in \mathbb{R}^{L \times N-1}$.

In term of the variables $\mathbf{y}, z$ and the constant matrix $\mathbf{W}$, the penalized maximum likelihood estimation problem in eq.(2) can be reformulated as:

$$\max_{\boldsymbol{\Omega} \succ \mathbf{0}} \left( \ \log(z - \mathbf{y}^{\mathrm{T}}\mathbf{W}^{-1}\mathbf{y}) - 2\mathbf{u}^{\mathrm{T}}\mathbf{y} - (v+\rho)z - 2\rho \left\| \mathbf{y} \right\|_1 - \tau \left\| \mathbf{A}\mathbf{y} - \mathbf{b} \right\|_1 \ \right) \tag{4}$$

where $\left\| \mathbf{A}\mathbf{y} - \mathbf{b} \right\|_1$ can be written in an extended form:

$$\left\| \mathbf{A}\mathbf{y} - \mathbf{b} \right\|_1 = \left\| \mathbf{D}_1 \mathbf{y} \right\|_1 + \left\| \mathbf{vec}(\mathbf{J}(\mathbf{D}_2) \circ (\mathbf{d}_3 \mathbf{y}^{\mathrm{T}} + \mathbf{D}_2 \mathbf{W})) \right\|_1 \tag{5}$$

Intuitively, the term $\left\| \mathbf{D}_1 \mathbf{y} \right\|_1$ penalizes differences across different rows of $\boldsymbol{\Omega}$ which affect only values in $\mathbf{y}$, while the term $\left\| \mathbf{vec}(\mathbf{J}(\mathbf{D}_2) \circ (\mathbf{d}_3 \mathbf{y}^{\mathrm{T}} + \mathbf{D}_2 \mathbf{W})) \right\|_1$ penalizes differences across different columns of $\boldsymbol{\Omega}$ which affect values of $\mathbf{y}$ as well as $\mathbf{W}$.

It can be shown that the precision matrix $\boldsymbol{\Omega}$ is positive definite since its Schur complement $z - \mathbf{y}^{\mathrm{T}}\mathbf{W}^{-1}\mathbf{y}$ is positive. By maximizing eq.(4) with respect to $z$, we get:

$$z - \mathbf{y}^{\mathrm{T}}\mathbf{W}^{-1}\mathbf{y} = \frac{1}{v+\rho} \tag{6}$$

and since $v > 0$ and $\rho > 0$, this implies that the Schur complement in eq.(6) is positive.

Maximization with respect to one variable at a time leads to a strictly convex, non-smooth, piecewise quadratic function. By replacing the optimal value for $z$ given by eq.(6) into the objective function in eq.(4), we get:

$$\min_{\mathbf{y} \in \mathbb{R}^{N-1}} \left( \ \tfrac{1}{2}\mathbf{y}^{\mathrm{T}}(v+\rho)\mathbf{W}^{-1}\mathbf{y} + \mathbf{u}^{\mathrm{T}}\mathbf{y} + \rho \left\| \mathbf{y} \right\|_1 + \tfrac{\tau}{2} \left\| \mathbf{A}\mathbf{y} - \mathbf{b} \right\|_1 \ \right) \tag{7}$$

Since the objective function in eq.(7) is non-smooth, its derivative is not continuous and therefore methods such as gradient descent cannot be applied. Although coordinate descent methods [5, 6] are suitable when only sparseness is enforced, they are not when local constancy is encouraged. As shown in [21], when penalizing an $\ell_1$-norm of differences, a coordinate descent algorithm can get stuck at sharp corners of the non-smooth optimization function; the resulting coordinates are stationary only under single-coordinate moves but not under diagonal moves involving two coordinates at a time.

For a discrete derivative operator $\mathbf{D}$ used in the penalized maximum likelihood estimation problem in eq.(2), it suffices to take directions involving either one variable $\mathbf{g} = (0, \ldots, 0, 1, 0, \ldots, 0)^{\mathrm{T}}$ or two spatially neighboring variables $\mathbf{g} = (0, \ldots, 0, 1, 0, \ldots, 0, 1, 0, \ldots, 0)^{\mathrm{T}}$ such that 1s appear in the position corresponding to the two neighbor variables. Finally, assuming an initial value $\mathbf{y}_0$ and a direction $\mathbf{g}$, the objective function in eq.(7) can be reduced to find $t$ in $\mathbf{y}(t) = \mathbf{y}_0 + t\mathbf{g}$ such that it minimizes:

$$\begin{array}{l}
\min_{t \in \mathbb{R}} \left( \tfrac{1}{2}pt^2 + qt + \sum_m r_m |t - s_m| \right) \\
p = (v+\rho)\mathbf{g}^{\mathrm{T}}\mathbf{W}^{-1}\mathbf{g} \quad , \quad q = ((v+\rho)\mathbf{W}^{-1}\mathbf{y}_0 + \mathbf{u})^{\mathrm{T}}\mathbf{g} \\
\mathbf{r} = \left[ \begin{array}{c} \rho|\mathbf{g}| \\ \tfrac{\tau}{2}|\mathbf{A}\mathbf{g}| \end{array} \right] \quad , \quad \mathbf{s} = \left[ \begin{array}{c} -\mathbf{diag}(\mathbf{g})^{-1}(\mathbf{y}_0) \\ -\mathbf{diag}(\mathbf{A}\mathbf{g})^{-1}(\mathbf{A}\mathbf{y}_0 - \mathbf{b}) \end{array} \right]
\end{array} \tag{8}$$

For simplicity of notation, we assume that $\mathbf{r}, \mathbf{s} \in \mathbb{R}^M$ use only non-zero entries of $\mathbf{g}$ and $\mathbf{A}\mathbf{g}$ on its definition in eq.(8). We sort and remove duplicate values in $\mathbf{s}$, and propagate changes to $\mathbf{r}$ by adding the entries corresponding to the duplicate values in $\mathbf{s}$. Note that these apparent modifications do not change the objective function, but they simplify its optimization. The resulting minimization problem in eq.(8) is convex, non-smooth and piecewise quadratic. Furthermore, since the objective function is quadratic on each interval $[-\infty; s_1], [s_1; s_2], \ldots, [s_{M-1}; s_M], [s_M; +\infty]$, it admits a closed form solution.

The coordinate-direction descent algorithm is presented in detail in Table 2. A careful implementation of the algorithm allows obtaining a time complexity of $O(KN^3)$ for $K$ iterations and $N$ variables, in which $\mathbf{W}^{-1}$, $\mathbf{W}^{-1}\mathbf{y}$ and $\mathbf{A}\mathbf{y}$ are updated at each iteration. In our experiments, the

| **Coordinate-direction descent algorithm** |
| :--- |

1. Given a dense sample covariance matrix $\widehat{\boldsymbol{\Sigma}}$, sparseness parameter $\rho$, local constancy parameter $\tau$ and a discrete derivative operator $\mathbf{D}$, find the precision matrix $\boldsymbol{\Omega} \succ \mathbf{0}$ that maximizes:

$$\log \det \boldsymbol{\Omega} - \langle \widehat{\boldsymbol{\Sigma}}, \boldsymbol{\Omega} \rangle - \rho \|\boldsymbol{\Omega}\|_1 - \tau \|\mathbf{D} \oslash \boldsymbol{\Omega}\|_1$$

2. Initialize $\boldsymbol{\Omega} = \mathbf{diag}(\widehat{\boldsymbol{\Sigma}})^{-1}$
3. For each iteration $1, \ldots K$ and each variable $1, \ldots, N$

   (a) Split $\boldsymbol{\Omega}$ into $\mathbf{W}, \mathbf{y}, z$ and $\widehat{\boldsymbol{\Sigma}}$ into $\mathbf{S}, \mathbf{u}, v$ as described in eq.(3)

   (b) Update $\mathbf{W}^{-1}$ by using the Sherman-Woodbury-Morrison formula (Note that when iterating from one variable to the next one, only one row and column change on matrix $\mathbf{W}$)

   (c) Transform local constancy regularization term from $\mathbf{D}$ into $\mathbf{A}$ and $\mathbf{b}$ as described in eq.(5)

   (d) Compute $\mathbf{W}^{-1}\mathbf{y}$ and $\mathbf{A}\mathbf{y}$

   (e) For each direction $\mathbf{g}$ involving either one variable or two spatially neighboring variables

       i. Find $t$ that minimizes eq.(8) in closed form

       ii. Update $\mathbf{y} \leftarrow \mathbf{y} + t\mathbf{g}$

       iii. Update $\mathbf{W}^{-1}\mathbf{y} \leftarrow \mathbf{W}^{-1}\mathbf{y} + t\mathbf{W}^{-1}\mathbf{g}$

       iv. Update $\mathbf{A}\mathbf{y} \leftarrow \mathbf{A}\mathbf{y} + t\mathbf{A}\mathbf{g}$

   (f) Update $z \leftarrow \frac{1}{v+\rho} + \mathbf{y}^{\mathrm{T}}\mathbf{W}^{-1}\mathbf{y}$

Table 2: Coordinate-direction descent algorithm for learning sparse and locally constant Gaussian graphical models.

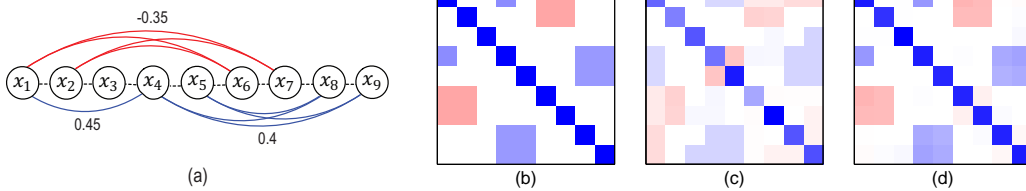

(a)        (b)        (c)        (d)

Figure 1: (a) Ground truth model on an open contour manifold. Spatial neighbors are connected with black dashed lines. Positive interactions are shown in blue, negative interactions in red. The model contains two locally constant interactions between $(x_1, x_2)$ and $(x_6, x_7)$, and between $(x_4, x_5)$ and $(x_8, x_9)$, a not locally constant interaction between $x_1$ and $x_4$, and an independent variable $x_3$; (b) colored precision matrix of the ground truth, red for negative entries, blue for positive entries; learnt structure from (c) small and (d) large datasets. Note that for large datasets all connections are correctly recovered.

algorithm converges quickly in usually $K = 10$ iterations. The polynomial dependency on the number of variables of $O(N^3)$ is expected since we cannot produce an algorithm faster than computing the inverse of the sample covariance in the case of an infinite sample.

Finally, in the spirit of [5], a method for reducing the size of the original problem is presented. Given a $P$-dimensional spatial neighborhood or manifold (e.g. $P = 1$ for silhouettes, $P = 2$ for a four-pixel neighborhood on 2D images, $P = 3$ for a six-pixel neighborhood on 3D images), the objective function in eq.(7) has the maximizer $\mathbf{y} = \mathbf{0}$ for variables on which $\|\mathbf{u}\|_\infty \leq \rho - P\tau$. Since this condition does not depend on specific entries in the iterative estimation of the precision matrix, this property can be used to reduce the size of the problem in advance by removing such variables.

## 5   Experimental Results

**Convergence to Ground Truth.** We begin with a small synthetic example to test the ability of the method for recovering the ground truth structure from data, in a complex scenario in which our method has to deal with both locally and not locally constant interactions as well as independent variables. The ground truth Gaussian graphical model is shown in Figure 1 and it contains 9 variables arranged in an open contour manifold.

In order to measure the closeness of the recovered models to the ground truth, we measure the Kullback-Leibler divergence, average precision (one minus the fraction of falsely included edges), average recall (one minus the fraction of falsely excluded edges) as well as the Frobenius norm between the recovered model and the ground truth. For comparison purposes, we picked two of the

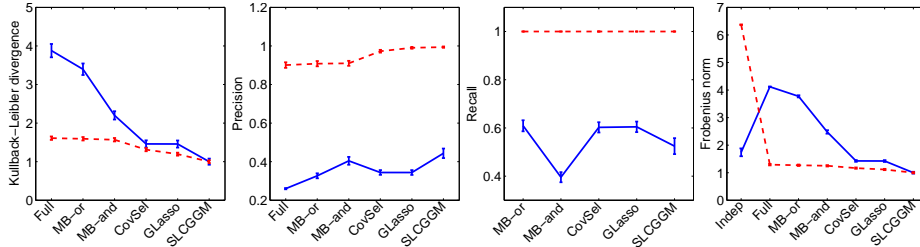

Figure 2: Kullback-Leibler divergence with respect to the best method, average precision, recall and Frobenius norm between the recovered model and the ground truth. Our method (SLCGGM) outperforms the fully connected model (Full), Meinshausen-Bühlmann approximation (MB-or, MB-and), covariance selection (CovSel), graphical lasso (GLasso) for small datasets (in blue solid line) and for large datasets (in red dashed line). The fully independent model (Indep) resulted in relative divergences of 2.49 for small and 113.84 for large datasets.

state-of-the-art structure learning techniques: covariance selection [5] and graphical lasso [6], since it has been shown theoretically and experimentally that they both converge to the maximum likelihood estimator. We also test the Meinshausen-Bühlmann approximation [4]. The fully connected as well as fully independent model are also included as baseline methods.

Two different scenarios are tested: small datasets of four samples, and large datasets of 400 samples. Under each scenario, 50 datasets are randomly generated from the ground truth Gaussian graphical model. It can be concluded from Figure 2 that our method outperforms the state-of-the-art structure learning techniques both for small and large datasets. This is due to the fact that the ground truth data contains locally constant interactions, and our method imposes a prior for local constancy. Although this is a complex scenario which also contains not locally constant interactions as well as an independent variable, our method can recover a more plausible model when compared to other methods. Note that even though other methods may exhibit a higher recall for small datasets, our method consistently recovers a better probability distribution.

A visual comparison of the ground truth versus the best recovered model by our method from small and large datasets is shown in Figure 1. The image shows the precision matrix in which red squares represent negative entries, while blue squares represent positive entries. There is very little difference between the ground truth and the recovered model from large datasets. Although the model is not fully recovered from small datasets, our technique performs better than the Meinshausen-Bühlmann approximation, covariance selection and graphical lasso in Figure 2.

**Real-World Datasets.** In the following experiments, we demonstrate the ability of our method to discover useful structures from real-world datasets. Datasets with a diverse nature of probabilistic relationships are included in our experiments: from cardiac MRI [22], our method recovers global deformation in the form of rotation and shrinking; from a walking sequence[1], our method finds the long range interactions between different parts; and from functional brain MRI [23], our method recovers functional interactions between different regions and discover differences in processing monetary rewards between cocaine addicted subjects versus healthy control subjects. Each dataset is also diverse in the type of spatial neighborhood: one-dimensional for silhouettes in a walking sequence, two-dimensional for cardiac MRI and three-dimensional for functional brain MRI.

**Generalization.** Cross-validation was performed in order to measure the generalization performance of our method in estimating the underlying distribution. Each dataset was randomly split into five sets. On each round, four sets were used for training and the remaining set was used for measuring the log-likelihood. Table 3 shows that our method consistently outperforms techniques that encourage sparsity only. This is strong evidence that datasets that are measured over a spatial manifold are locally constant, as well as that our method is a good regularization technique that avoids over-fitting and allows for better generalization. Another interesting fact is that for the brain MRI dataset, which is high dimensional and contains a small number of samples, the model that assumes full independence performed better than the Meinshausen-Bühlmann approximation, covariance selection and graphical lasso. Similar observations has been already made in [24, 25] where it was found that assuming independence often performs better than learning dependencies among variables.

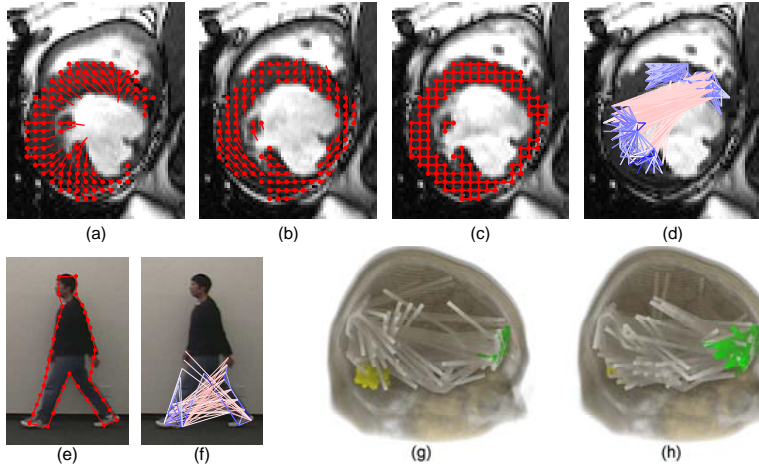

Figure 3: Real-world datasets: cardiac MRI displacement (a) at full contraction and (b) at full expansion, (c) 2D spatial manifold and (d) learnt structure, which captures contraction and expansion (in red), and similar displacements between neighbor pixels (in blue); (e) silhouette manifold and (f) learnt structure from a manually labeled walking sequence, showing similar displacements from each independent leg (in blue) and opposite displacements between both legs as well as between hands and feet (in red); and structures learnt from functional brain MRI in a monetary reward task for (g) drug addicted subjects with more connections in the cerebellum (in yellow) versus (h) control subjects with more connections in the prefrontal cortex (in green).

| Method | Synthetic | Cardiac MRI | Walking Sequence | Brain MRI Drug-addicted | Brain MRI Control |
|---|---|---|---|---|---|
| Indep | -6428.23 | -5150.58 | -12957.72 | -324724.24 | -302729.54 |
| MB-and | -5595.87* | -5620.45 | -12542.15 | -418605.02 | -317034.67 |
| MB-or | -5595.13* | -4135.98* | -11317.24 | -398725.04 | -298186.66 |
| CovSel | -5626.32 | -5044.41 | -12051.51 | -409402.60 | -300829.98 |
| GLasso | -5625.79 | -5041.52 | -12035.50 | -413176.45 | -305307.25 |
| SLCGGM | **-5623.52** | **-4017.56** | **-10718.62** | **-297318.61** | **-278678.35** |

Table 3: Cross-validated log-likelihood on the testing set. Our method (SLCGGM) outperforms the Meinshausen-Bühlmann approximation (MB-and, MB-or), covariance selection (CovSel), graphical lasso (GLasso) and the fully independent model (Indep). Values marked with an asterisk are not statistically significantly different from our method.

## 6 Conclusions and Future Work

In this paper, we proposed local constancy for Gaussian graphical models, which encourages finding probabilistic connectivities between two close or distant clusters of variables, instead of between isolated variables. We introduced an $\ell_1$-norm penalty for local constancy into a strictly convex maximum likelihood estimation. Furthermore, we proposed an efficient optimization algorithm and proved that our method guarantees positive definiteness of the estimated precision matrix. We tested the ability of our method to recover the ground truth structure from data, in a complex scenario with locally and not locally constant interactions as well as independent variables. We also tested the generalization performance of our method in a wide range of complex real-world datasets with a diverse nature of probabilistic relationships as well as neighborhood type.

There are several ways of extending this research. Methods for selecting regularization parameters for sparseness and local constancy need to be further investigated. Although the positive definiteness properties of the precision matrix as well as the optimization algorithm still hold when including operators such as the Laplacian for encouraging smoothness, benefits of such a regularization approach need to be analyzed. In practice, our technique converges in a small number of iterations, but a more precise analysis of the rate of convergence needs to be performed. Finally, model selection consistency when the number of samples grows to infinity needs to be proved.

## Acknowledgments

This work was supported in part by NIDA Grant 1 R01 DA020949-01 and NSF Grant CNS-0721701

## Footnotes

[1] Human Identification at a Distance dataset `http://www.cc.gatech.edu/cpl/projects/hid/`

# References

[1] D. Crandall, P. Felzenszwalb, and D. Huttenlocher. Spatial priors for part-based recognition using statistical models. *IEEE Conf. Computer Vision and Pattern Recognition*, 2005.

[2] P. Felzenszwalb and D. Huttenlocher. Pictorial structures for object recognition. *International Journal of Computer Vision*, 2005.

[3] L. Gu, E. Xing, and T. Kanade. Learning GMRF structures for spatial priors. *IEEE Conf. Computer Vision and Pattern Recognition*, 2007.

[4] N. Meinshausen and P. Bühlmann. High dimensional graphs and variable selection with the lasso. *The Annals of Statistics*, 2006.

[5] O. Banerjee, L. El Ghaoui, A. d'Aspremont, and G. Natsoulis. Convex optimization techniques for fitting sparse Gaussian graphical models. *International Conference on Machine Learning*, 2006.

[6] J. Friedman, T. Hastie, and R. Tibshirani. Sparse inverse covariance estimation with the graphical lasso. *Biostatistics*, 2007.

[7] M. Yuan and Y. Lin. Model selection and estimation in the Gaussian graphical model. *Biometrika*, 2007.

[8] E. Levina, A. Rothman, and J. Zhu. Sparse estimation of large covariance matrices via a nested lasso penalty. *The Annals of Applied Statistics*, 2008.

[9] J. Duchi, S. Gould, and D. Koller. Projected subgradient methods for learning sparse Gaussians. *Uncertainty in Artificial Intelligence*, 2008.

[10] V. Mansinghka, C. Kemp, J. Tenenbaum, and T. Griffiths. Structured priors for structure learning. *Uncertainty in Artificial Intelligence*, 2006.

[11] S. Lauritzen. *Graphical Models*. Oxford Press, 1996.

[12] A. Dempster. Covariance selection. *Biometrics*, 1972.

[13] R. Tibshirani. Regression shrinkage and selection via the lasso. *Journal of the Royal Statistical Society*, 1996.

[14] O. Banerjee, L. El Ghaoui, and A. d'Aspremont. Model selection through sparse maximum likelihood estimation for multivariate Gaussian or binary data. *Journal of Machine Learning Research*, 2008.

[15] S. Lee, V. Ganapathi, and D. Koller. Efficient structure learning of Markov networks using $\ell_1$-regularization. *Advances in Neural Information Processing Systems*, 2006.

[16] M. Schmidt, A. Niculescu-Mizil, and K. Murphy. Learning graphical model structure using $\ell_1$-regularization paths. *AAAI Conf. Artificial Intelligence*, 2007.

[17] M. Schmidt, K. Murphy, G. Fung, and R. Rosales. Structure learning in random fields for heart motion abnormality detection. *IEEE Conf. Computer Vision and Pattern Recognition*, 2008.

[18] M. Wainwright, P. Ravikumar, and J. Lafferty. High dimensional graphical model selection using $\ell_1$-regularized logistic regression. *Advances in Neural Information Processing Systems*, 2006.

[19] R. Tibshirani, M. Saunders, S. Rosset, J. Zhu, and K. Knight. Sparsity and smoothness via the fused lasso. *Journal of the Royal Statistical Society*, 2005.

[20] S. Boyd and L. Vandenberghe. *Convex Optimization*. Cambridge University Press, 2006.

[21] J. Friedman, T. Hastie, H. Höfling, and R. Tibshirani. Pathwise coordinate optimization. *The Annals of Applied Statistics*, 2007.

[22] J. Deux, A. Rahmouni, and J. Garot. Cardiac magnetic resonance and 64-slice cardiac CT of lipomatous metaplasia of chronic myocardial infarction. *European Heart Journal*, 2008.

[23] R. Goldstein, D. Tomasi, N. Alia-Klein, L. Zhang, F. Telang, and N. Volkow. The effect of practice on a sustained attention task in cocaine abusers. *NeuroImage*, 2007.

[24] P. Domingos and M. Pazzani. On the optimality of the simple Bayesian classifier under zero-one loss. *Machine Learning*, 1997.

[25] N. Friedman, D. Geiger, and M. Goldszmidt. Bayesian network classifiers. *Machine Learning*, 1997.

